# Modeling User Rating Profiles For Collaborative Filtering

**Benjamin Marlin**
Department of Computer Science
University of Toronto
Toronto, ON, M5S 3H5, CANADA
marlin@cs.toronto.edu

## Abstract

In this paper we present a generative latent variable model for rating-based collaborative filtering called the User Rating Profile model (URP). The generative process which underlies URP is designed to produce complete user rating profiles, an assignment of one rating to each item for each user. Our model represents each user as a mixture of *user attitudes*, and the mixing proportions are distributed according to a Dirichlet random variable. The rating for each item is generated by selecting a user attitude for the item, and then selecting a rating according to the *preference pattern* associated with that attitude. URP is related to several models including a multinomial mixture model, the aspect model [7], and LDA [1], but has clear advantages over each.

## 1 Introduction

In rating-based collaborative filtering, users express their preferences by explicitly assigning ratings to items that they have accessed, viewed, or purchased. We assume a set of $N$ users $\{1, ..., N\}$, a set of $M$ items $\{1, ..., M\}$, and a set of $V$ discrete rating values $\{1, ..., V\}$. In the natural case where each user has at most one rating $r_y^u$ for each item $y$, the ratings for each user form a vector with one component per item. Of course, the values of some components are not known. We refer to user $u$'s rating vector as their rating profile denoted $\mathbf{r}^u$.

Rating prediction is the elementary task performed with rating-based data. Given a particular item and user, the goal is to predict the user's true rating for the item in question. Early work on rating prediction focused on neighborhood-based methods such as the GroupLens algorithm [9]. Personalized recommendations can be generated for any user by first predicting ratings for all items the user has not rated, and recommending items with the highest predicted ratings. The capability to predict ratings has other interesting applications. Rating predictions can be incorporated with content-based scores to create a preference augmented search procedure [4]. Rating prediction also facilitates an active approach to collaborative filtering using expected value of information. In such a framework the predicted rating of each item is interpreted as its expected utility to the user [2].

In order to gain the maximum advantage from the expressive power of ratings, a probabilistic model must enable the calculation of the distribution over ratings, and thus the calculation of predicted ratings. A handful of such models exist including the multinomial mixture model shown in figure 3, and the aspect model shown in figure 1 [7]. As latent variable models, both the aspect model and the multinomial mixture model have an intuitive appeal. They can be interpreted as decomposing user preferences profiles into a set of typical preference patterns, and the degree to which each user participates in each preference pattern. The settings of the latent variable are casually referred to as *user attitudes*. The multinomial mixture model constrains all users to have the same prior distribution over user attitudes, while the aspect model allows each user to have a different prior distribution over user attitudes. The added flexibility of the aspect model is quite attractive, but the interpretation of the distribution over user attitudes as parameters instead of random variables induces several problems.[1]First, the aspect model lacks a principled, maximum likelihood inference procedure for novel user profiles. Second the number of parameters in the model grows linearly with the number of users in the data set.

Recent research has seen the proposal of several generative latent variable models for discrete data, including Latent Dirichlet Allocation [1] shown in figure 2, and multinomial PCA (a generalization of LDA to priors other than Dirichlet) [3]. LDA and mPCA were both designed with co-occurrence data in mind (word-document pairs). They can only be applied to rating data if the data is first processed into user-item pairs using some type of thresholding operation on the rating values. These models can then be used to generate recommendations; however, they can not be used to infer a distribution over ratings of items, or to predict the ratings of items.

The contribution of this paper is a new generative, latent variable model that views rating-based data at the level of user rating profiles. The URP model incorporates proper generative semantics at the user level that are similar to those used in LDA and mPCA, while the inner workings of the model are designed specifically for rating profiles. Like the aspect model and the multinomial mixture model, the URP model can be interpreted in terms of decomposing rating profiles into typical preference patterns, and the degree to which each user participates in each pattern. In this paper we describe the URP model, give model fitting and initialization procedures, and present empirical results for two data sets.

## 2    The User Rating Profile Model

The graphical representation of the aspect, LDA, multinomial mixture, and URP models are shown in figures 1 through 4. In all models $U$ is a user index, $Y$ is an item index, $Z$ is a user attitude, $Z_y$ is the user attitude responsible for item $y$, $R$ is a rating value, $R_y$ is a rating value for item $Y$, and $\beta_{vyz}$ is a multinomial parameter giving $P(R_y = v | Z_y = z)$. In the aspect model $\theta$ is a set of multinomial parameters where $\theta_z^u$ represents $P(Z = z | U = u)$. The number of these parameters obviously grows as the number of training users is increased. In the mixture of multinomials model $\theta$ is a single distribution over user attitudes where $\theta_z$ represents $P(Z = z)$. This gives the multinomial mixture model correct, yet simplistic, generative semantics at the user level. In both LDA and URP $\theta$ is not a parameter, but a Dirichlet random variable with parameter $\alpha$. A unique $\theta$ is sampled for each user where $\theta_z$ gives

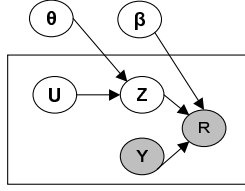

Figure 1: Aspect Model

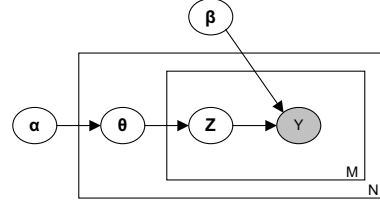

Figure 2: LDA Model

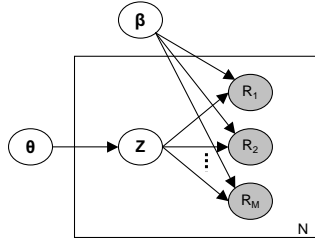

Figure 3: Multinomial Mixture Model

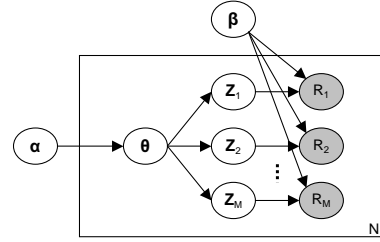

Figure 4: URP Model

$P(Z = z)$ for that user. This gives URP much more powerful generative semantics at the user level than the multinomial mixture model. As with LDA, URP could be generalized to use any continuous distribution on the simplex, but in this case the Dirichlet leads to efficient prediction equations. Note that the bottom level of the LDA model consists of an item variable $Y$, and ratings do not come into LDA at any point.

The probability of observing a given user rating profile $\mathbf{r}^u$ under the URP model is shown in equation 1 where we define $\delta(\mathbf{r}^u_y, v)$ to be equal to 1 if user $u$ assigned rating $v$ to item $y$, and 0 otherwise. Note that we assume unspecified ratings are missing at random. As in LDA, the Dirichlet prior renders the computation of the posterior distribution $p(\theta, \mathbf{z}|\mathbf{r}^u, \alpha, \beta) = P(\theta, \mathbf{z}, \mathbf{r}^u|\alpha, \beta)/P(\mathbf{r}^u|\alpha, \beta)$ intractable.

$$P(\mathbf{r}^u|\alpha, \beta) = \int_\theta P(\theta|\alpha) \prod_{y=1}^M \prod_{v=1}^V \left( \sum_{z=1}^K P(Z_y = z|\theta) P(R_y = v|Z_y = z, \beta) \right)^{\delta(r^u_y, v)} d\theta \tag{1}$$

## 3    Parameter Estimation

The procedure we use for parameter estimation is a variational expectation maximization algorithm based on free energy maximization. As with LDA, other methods including expectation propagation could be applied. We choose to apply a fully factored variational $q$-distribution as shown in equation 2. We define $q(\theta|\gamma^u)$ to be a Dirichlet distribution with Dirichlet parameters $\gamma^u_z$, and $q(Z_y|\phi^u_y)$ to be a multinomial distribution with parameters $\phi^u_{zy}$.

$$P(\theta, \mathbf{z}|\alpha, \beta, \mathbf{r}^u) \approx q(\theta, \mathbf{z}|\gamma^u, \phi^u) = q(\theta|\gamma^u) \prod_{y=1}^M q(Z_y = z_y|\phi^u_y) \tag{2}$$

A per-user free energy function $F[\gamma^u, \phi^u, \alpha, \beta]$ provides a variational lower bound on the log likelihood $\log p(\mathbf{r}^u | \alpha, \beta)$ of a single user rating profile. The sum of the per-user free energy functions $F[\gamma^u, \phi^u, \alpha, \beta]$ yields the total free energy function $F[\gamma, \phi, \alpha, \beta]$, which is a lower bound on the log likelihood of a complete data set of user rating profiles. The variational and model parameter updates are obtained by expanding $F[\gamma, \phi, \alpha, \beta]$ using the previously described distributions, and maximizing the result with respect to $\gamma^u$, $\phi^u$, $\alpha$ and $\beta$. The variational parameter updates are shown in equations 3, and 4. $\Psi$ denotes the first derivative of the log gamma function, also know as the digamma or psi function.

$$\phi^u_{zy} \quad \propto \quad \prod_{v=1}^{V} \beta_{vyz}{}^{\delta(r^u_y, v)} \exp(\Psi(\gamma^u_z) - \Psi(\textstyle\sum_{j=1}^{k} \gamma^u_j)) \tag{3}$$

$$\gamma^u_z \quad = \quad \alpha_z + \sum_{y=1}^{M} \phi^u_{zy} \tag{4}$$

By iterating the the variational updates with fixed $\alpha$ and $\beta$ for a particular user, we are guaranteed to reach a local maximum of the per-user free energy $F[\gamma^u, \phi^u, \alpha, \beta]$. This iteration is a well defined approximate inference procedure for the URP model.

The model multinomial update has a closed form solution as shown in equation 5. This is not the case for the model Dirichlet $\alpha$ due to coupling of its parameters. However, Minka has proposed two iterative methods for estimating a Dirichlet distribution from probability vectors that can be used here. We give Minka's fixed-point iteration in equations 6 and 7, which yields very similar results compared to the alternative Newton iteration. Details for both procedures including the inversion of the digamma function may be found in [8].

$$\beta_{vyz} \quad \propto \quad \sum_{u=1}^{N} \phi^u_{zy} \delta(r^u_y, v) \tag{5}$$

$$\Psi(\alpha_z) \quad = \quad \Psi(\sum_{z=1}^{K} \alpha_z) + 1/N(\sum_{u=1}^{N} \Psi(\gamma^u_z) - \Psi(\sum_{u=1}^{N} \gamma^u_z)) \tag{6}$$

$$\alpha_z \quad = \quad \Psi^{-1}(\Psi(\alpha_z)) \tag{7}$$

## 4 Model Fitting and Initialization

We give a variational expectation maximization procedure for model fitting in this section as well as an initialization method that has proved to be very effective for the URP model. Lastly, we discuss stopping criteria used for the EM iterations.

### 4.1 Model Fitting

The variational inference procedure should be run to convergence to insure a maximum likelihood solution. However, if we are satisfied with simply increasing the free energy at each step, other fitting procedures are possible. In general, the number of steps of variational inference can be determined by a user dependant heuristic function $H(u)$. Buntine uses a single step of variational inference for each user to fit the mPCA model. At the other end of the spectrum, Blei et al. select a sufficient

number of steps to achieve convergence when fitting the LDA model. Empirically, we have found that simple linear functions, of the number of ratings in each user profile provide a good heuristic. The details of the fitting procedure are given below.

**E-Step:**

1. For all users $u$

2.      For $h = 0$ to $H(u)$

3.         $\phi_{zy}^{u} \propto \prod_{v=1}^{V} \beta_{ryz}{}^{\delta(r_{y}^{u}, v)} \exp(\Psi(\gamma_{z}^{u}) - \Psi(\sum_{j=1}^{k} \gamma_{j}^{u}))$

4.         $\gamma_{z}^{u} = \alpha_{z} + \sum_{y=1}^{M} \phi_{zy}^{u}$

**M-Step:**

1. For each $v, y, z$ set $\beta_{vyz} \propto \sum_{u=1}^{N} \phi_{zyv}^{u} \delta(r_{y}^{u}, v)$.

2. While not converged

3.      $\Psi(\alpha_{z}) = \Psi(\sum_{z=1}^{K} \alpha_{z}) + 1/N(\sum_{u=1}^{N} \Psi(\gamma_{z}^{u}) - \Psi(\sum_{u=1}^{N} \gamma_{z}^{u}))$

4.      $\alpha_{z} = \Psi^{-1}(\Psi(\alpha_{z}))$

## 4.2 Initialization and Early Stopping

Fitting the URP model can be quite difficult starting from randomly initialized parameters. The initialization method we have adopted is to partially fit a multinomial mixture model with the same number of user attitudes as the URP model. Fitting the multinomial mixture model for a small number of EM iterations yields a set of multinomial distributions encoded by $\beta'$, as well as a single multinomial distribution over user attitudes encoded by $\theta'$. To initialize the URP model we set $\beta = \beta'$, $\alpha = \kappa\theta'$ where $\kappa$ is a positive constant. Letting $\kappa = 1$ appears to give good results in practice.

Normally EM is run until the bound on log likelihood converges, but this tends to lead to over fitting in some models including the aspect model. To combat this problem Hofmann suggests using early stopping of the EM iteration [7]. We implemented early stopping for all models using a separate validation set to allow for a fair comparison.

## 5 Prediction

The primary task for any model applied to the rating-based collaborative filtering problem is to predict ratings for the items a user has not rated, based on the ratings the user has specified. Assume we have a user $u$ with rating profile $\mathbf{r}^{u}$, and we wish to predict the user's rating $r_{y}^{u}$ for an unrated item $y$. The distribution over ratings for the item $y$ can be calculated using the model as follows:

$$P(R_{y} = v | \mathbf{r}^{u}) = \int_{\theta} \sum_{z} P(R_{y} = v | Z_{y} = z) P(Z_{y} = z | \theta) P(\theta | \mathbf{r}^{u}) d\theta \qquad (8)$$

This quantity may look quite difficult to compute, but by interchanging the sum and integral, and appealing to our variational approximation $q(\theta | \gamma^{u}) \approx P(\theta | \mathbf{r}^{u})$ we obtain an expression in terms of the model and variational parameters.

$$p(R_y = v|\mathbf{r}^u) = \sum_{z=1}^{K} \beta_{vyz} \frac{\gamma_z^u}{\sum_{j=1}^{K} \gamma_j^u} \tag{9}$$

To compute $P(R_y = v|\mathbf{r}^u)$ according to equation 9 given the model parameters $\alpha$ and $\beta$, it is necessary to apply our variational inference procedure to compute $\gamma^u$. However, this only needs to be done once for each user in order to predict all unknown ratings in the user's profile. Given the distribution $P(R_y|\mathbf{r}^u)$, various rules can be used to compute the predicted rating. One could predict the rating with maximal probability, predict the expected rating, or predict the median rating. Of course, each of these prediction rules minimizes a different prediction error measure. In particular, median prediction minimizes the mean absolute error and is the prediction rule we use in our experiments.

## 6  Experimentation

We consider two different experimental procedures that test the predictive ability of a rating-based collaborative filtering method. The first is a weak generalization all-but-1 experiment where one of each user's ratings is held out. The model is then trained on the remaining observed ratings and tested on the held out ratings. This experiment is designed to test the ability of a method to generalize to other items rated by the users it was trained on.

We introduce a second experimental protocol for testing a stronger form of generalization. The model is first trained using all ratings from a set of training users. Once the model is trained, an all-but-1 experiment is performed using a separate set of test users. This experiment is designed to test the ability of the model to generalize to novel user profiles.

Two different base data sets were used in the experiments. The well known Each-Movie data set, and the recently released million rating MovieLens data set. Both data sets were filtered to contain users with at least 20 ratings. EachMovie was filtered to remove movies with less than 2 ratings leaving 1621 movies. The Movie-Lens data was similarly filtered leaving 3592 movies. The EachMovie training sets contained 30000 users while the test sets contained 5000 users. The MovieLens training sets contained 5000 users while the test sets contained 1000 users. The EachMovie rating scale is from 0 to 5, while the MovieLens rating scale is from 1 to 5.

Both types of experiment were performed for a range of numbers of user attitudes. For each model and number of user attitudes, each experiment was repeated on three different random partitions of each base data set into known ratings, held out ratings, validation ratings, training users and testing users. In the weak generalization experiments the aspect, multinomial mixture, and URP models were tested. In the strong generalization experiments only the multinomial mixture and URP models were tested since a trained aspect model can not be applied to new user profiles. Also recall that LDA and mPCA can not be used for rating prediction so they are not be tested in these experiments. We provide results obtained with a best-$K$-neighbors version of the GroupLens method for various values of $K$ as a baseline method.

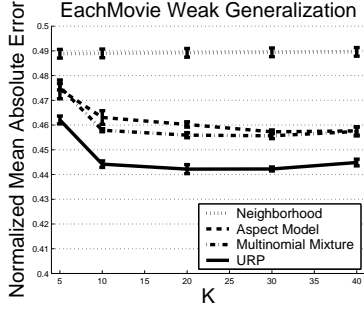

Figure 5: EachMovie weak generalization.

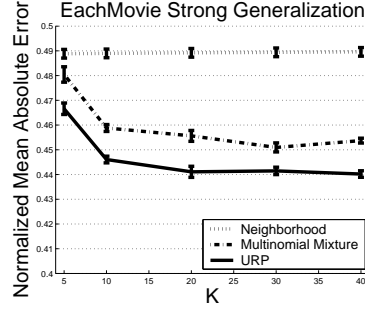

Figure 6: EachMovie strong generalization.

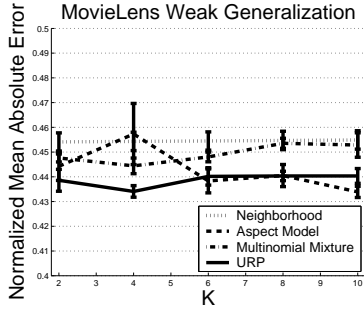

Figure 7: MovieLens weak generalization.

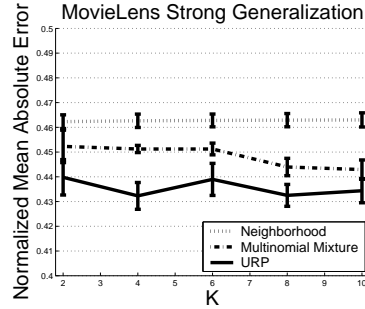

Figure 8: MovieLens strong generalization.

## 7 Results

Results are reported in figures 5 through 8 in terms of normalized mean absolute error (NMAE). We define our NMAE to be the standard MAE normalized by the the expected value of the MAE assuming uniformly distributed rating values and rating predictions. For the EachMovie dataset $E[MAE]$ is $1.9\bar{4}$, and for the MovieLens data set it is 1.6. Note that our definition of NMAE differs from that used by Goldberg et al. [6]. Goldberg et al. take the normalizer to be the difference between the minimum and maximum ratings, which means most of the error scale corresponds to performing much worse than random.

In both the weak and strong generalization experiments using the EachMovie data set, the URP model performs significantly better than the other methods, and obtains the lowest prediction error. The results obtained from the MovieLens data set do not show the same clean trends as the EachMovie data set for the weak generalization experiment. The smaller size of MovieLens data set seems to cause URP to over fit for larger values of $K$, thus increasing its test error. Nevertheless, the lowest error attained by URP is not significantly different than that obtained by the aspect model. In the strong generalization experiment the URP model again out performs the other methods.

# 8 Conclusions

In this paper we have presented the URP model for rating-based collaborative filtering. Our model combines the intuitive appeal of the multinomial mixture and aspect models, with the strong high level generative semantics of LDA and mPCA. As a result of being specially designed for collaborative filtering, our model also contains unique rating profile generative semantics not found in LDA or mPCA. This gives URP the capability to operate directly on ratings data, and to efficiently predict all missing ratings in a user profile. This means URP can be applied to recommendation, as well as many other tasks based on rating prediction.

We have empirically demonstrated on two different data sets that the weak generalization performance of URP is at least as good as that of the aspect and multinomial mixture models. For online applications where it is impractical to refit the model each time a rating is supplied by a user, the result of interest is strong generalization performance. The aspect model can not be applied in a principled manner in such a scenario, and we see that URP outperforms the other methods by a significant margin.

### Acknowledgments

We thank the Compaq Computer Corporation for the use of the EachMovie data set, and the GroupLens Research Group at the University of Minnesota for use of the MovieLens data set. Many thanks go to Rich Zemel for helpful comments and numerous discussions about this work.

## Footnotes

[1]Girolami and Kabán have recently shown that a co-occurrence version of the aspect model can be interpreted as a MAP/ML estimated LDA model under a uniform Dirichlet prior [5]. Essentially the same relationship holds between the aspect model for ratings shown in figure 1, and the URP model.

# References

[1] D. Blei, A. Ng, and M. Jordan. Latent Dirichlet allocation. *Journal of Machine Learning Research*, 3:993–1022, Jan. 2003.

[2] C. Boutilier, R. S. Zemel, and B. Marlin. Active collaborative filtering. In *Proceedings of the Nineteenth Annual Conference on Uncertainty in Artificial Intelligence*, pages 98–106, 2003.

[3] W. Buntine. Variational extensions to EM and multinomial PCA. In *Proceedings of the European Conference on Machine Learning*, 2002.

[4] M. Claypool, A. Gokhale, T. Miranda, P. Murnikov, D. Netes, and M. Sartin. Combining content-based and collaborative filters in an online newspaper. In *Proceedings of ACM SIGIR Workshop on Recommender Systems*, 1999.

[5] M. Girolami and A. Kabán. On an equivalence between PLSI and LDA. In *Proceedings of the ACM Conference on Research and Development in Information Retrieval*, pages 433–434, 2003.

[6] K. Goldberg, T. Roeder, D. Gupta, and C. Perkins. Eigentaste: A constant time collaborative filtering algorithm. *Information Retrieval Journal*, 4(2):133–151, July 2001.

[7] T. Hofmann. Learning What People (Don't) Want. In *Proceedings of the European Conference on Machine Learning*, 2001.

[8] T. Minka. Estimating a Dirichlet Distribution. *Unpublished*, 2003.

[9] P. Resnick, N. Iacovou, M. Suchak, P. Bergstorm, and J. Riedl. GroupLens: An Open Architecture for Collaborative Filtering of Netnews. In *Proceedings of ACM 1994 Conference on Computer Supported Cooperative Work*, pages 175–186, Chapel Hill, North Carolina, 1994. ACM.
